# A Bayesian Spatial Scan Statistic

**Daniel B. Neill   Andrew W. Moore**
School of Computer Science
Carnegie Mellon University
Pittsburgh, PA 15213
{neill,awm}@cs.cmu.edu

**Gregory F. Cooper**
Center for Biomedical Informatics
University of Pittsburgh
Pittsburgh, PA 15213
gfc@cbmi.pitt.edu

## Abstract

We propose a new Bayesian method for spatial cluster detection, the "Bayesian spatial scan statistic," and compare this method to the standard (frequentist) scan statistic approach. We demonstrate that the Bayesian statistic has several advantages over the frequentist approach, including increased power to detect clusters and (since randomization testing is unnecessary) much faster runtime. We evaluate the Bayesian and frequentist methods on the task of prospective disease surveillance: detecting spatial clusters of disease cases resulting from emerging disease outbreaks. We demonstrate that our Bayesian methods are successful in rapidly detecting outbreaks while keeping number of false positives low.

## 1   Introduction

Here we focus on the task of *spatial cluster detection*: finding spatial regions where some quantity is significantly higher than expected. For example, our goal may be to detect clusters of disease cases, which may be indicative of a naturally occurring epidemic (e.g. influenza), a bioterrorist attack (e.g. anthrax release), or an environmental hazard (e.g. radiation leak). [1] discusses many other applications of cluster detection, including mining astronomical data, medical imaging, and military surveillance. In all of these applications, we have two main goals: to identify the locations, shapes, and sizes of potential clusters, and to determine whether each potential cluster is more likely to be a "true" cluster or simply a chance occurrence. Thus we compare the null hypothesis $H_0$ of no clusters against some set of alternative hypotheses $H_1(S)$, each representing a cluster in some region or regions $S$. In the standard frequentist setting, we do this by significance testing, computing the $p$-values of potential clusters by randomization; here we propose a Bayesian framework, in which we compute posterior probabilities of each potential cluster.

Our primary motivating application is *prospective disease surveillance*: detecting spatial clusters of disease cases resulting from a disease outbreak. In this application, we perform surveillance on a daily basis, with the goal of finding emerging epidemics as quickly as possible. For this task, we are given the number of cases of some given syndrome type (e.g. respiratory) in each spatial location (e.g. zip code) on each day. More precisely, we typically cannot measure the actual number of cases, and instead rely on related observable quantities such as the number of Emergency Department visits or over-the-counter drug sales. We must then detect those increases which are indicative of emerging outbreaks, as close to the start of the outbreak as possible, while keeping the number of false positives low. In biosurveillance of disease, every hour of earlier detection can translate into thousands of lives saved by more timely administration of antibiotics, and this has led to widespread interest in systems for the rapid and automatic detection of outbreaks.

In this spatial surveillance setting, each day we have data collected for a set of discrete spatial locations $s_i$. For each location $s_i$, we have a *count* $c_i$ (e.g. number of disease cases), and an underlying *baseline* $b_i$. The baseline may correspond to the underlying *population* at risk, or may be an estimate of the expected value of the count (e.g. derived from the time series of previous count data). Our goal, then, is to find if there is any spatial region $S$ (set of locations $s_i$) for which the counts are significantly higher than expected, given the baselines. For simplicity, we assume here (as in [2]) that the locations $s_i$ are aggregated to a uniform, two-dimensional, $N \times N$ grid $G$, and we search over the set of rectangular regions $S \subseteq G$. This allows us to search both compact and elongated regions, allowing detection of elongated disease clusters resulting from dispersal of pathogens by wind or water.

## 1.1 The frequentist scan statistic

One of the most important statistical tools for cluster detection is Kulldorff's *spatial scan statistic* [3-4]. This method searches over a given set of spatial regions, finding those regions which maximize a likelihood ratio statistic and thus are most likely to be generated under the alternative hypothesis of clustering rather than the null hypothesis of no clustering. Randomization testing is used to compute the *p*-value of each detected region, correctly adjusting for multiple hypothesis testing, and thus we can both identify potential clusters and determine whether they are significant. Kulldorff's framework assumes that counts $c_i$ are Poisson distributed with $c_i \sim \text{Po}(qb_i)$, where $b_i$ represents the (known) census population of cell $s_i$ and $q$ is the (unknown) underlying disease rate. Then the goal of the scan statistic is to find regions where the disease rate is higher inside the region than outside. The statistic used for this is the likelihood ratio $F(S) = \frac{P(Data \, | \, H_1(S))}{P(Data \, | \, H_0)}$, where the null hypothesis $H_0$ assumes a uniform disease rate $q = q_{all}$. Under $H_1(S)$, we assume that $q = q_{in}$ for all $s_i \in S$, and $q = q_{out}$ for all $s_i \in G - S$, for some constants $q_{in} > q_{out}$. From this, we can derive an expression for $F(S)$ using maximum likelihood estimates of $q_{in}$, $q_{out}$, and $q_{all}$: $F(S) = (\frac{C_{in}}{B_{in}})^{C_{in}} (\frac{C_{out}}{B_{out}})^{C_{out}} (\frac{C_{all}}{B_{all}})^{-C_{all}}$, if $\frac{C_{in}}{B_{in}} > \frac{C_{out}}{B_{out}}$, and $F(S) = 1$ otherwise. In this expression, we have $C_{in} = \sum_S c_i$, $C_{out} = \sum_{G-S} c_i$, $C_{all} = \sum_G c_i$, and similarly for the baselines $B_{in} = \sum_S b_i$, $B_{out} = \sum_{G-S} b_i$, and $B_{all} = \sum_G b_i$.

Once we have found the highest scoring region $S^* = \arg\max_S F(S)$ of grid $G$, and its score $F^* = F(S^*)$, we must still determine the statistical significance of this region by randomization testing. To do so, we randomly create a large number $R$ of replica grids by sampling under the null hypothesis $c_i \sim \text{Po}(q_{all} b_i)$, and find the highest scoring region and its score for each replica grid. Then the *p*-value of $S^*$ is $\frac{R_{beat}+1}{R+1}$, where $R_{beat}$ is the number of replicas $G'$ with $F^*$ higher than the original grid. If this *p*-value is less than some threshold (e.g. 0.05), we can conclude that the discovered region is unlikely to have occurred by chance, and is thus a significant spatial cluster; otherwise, no significant clusters exist.

The frequentist scan statistic is a useful tool for cluster detection, and is commonly used in the public health community for detection of disease outbreaks. However, there are three main disadvantages to this approach. First, it is difficult to make use of any prior information that we may have, for example, our prior beliefs about the size of a potential outbreak and its impact on disease rate. Second, the accuracy of this technique is highly dependent on the correctness of our maximum likelihood parameter estimates. As a result, the model is prone to parameter overfitting, and may lose detection power in practice because of model misspecification. Finally, the frequentist scan statistic is very time consuming, and may be computationally infeasible for large datasets. A naive approach requires searching over all rectangular regions, both for the original grid and for each replica grid. Since there are $O(N^4)$ rectangles to search for an $N \times N$ grid, the total computation time is $O(RN^4)$, where $R = 1000$ is a typical number of replications. In past work [5, 2, 6], we have shown how to reduce this computation time by a factor of 20-2000x through use of the "fast spatial scan" algorithm; nevertheless, we must still perform this faster search both for the original grid and for each replica.

We propose to remedy these problems through the use of a Bayesian spatial scan statistic. First, our Bayesian model makes use of prior information about the likelihood, size, and impact of an outbreak. If these priors are chosen well, we should achieve better detection power than the frequentist approach. Second, the Bayesian method uses a *marginal likelihood* approach, averaging over possible values of the model parameters $q_{in}$, $q_{out}$, and $q_{all}$, rather than relying on maximum likelihood estimates of these parameters. This makes the model more flexible and less prone to overfitting, and reduces the potential impact of model misspecification. Finally, under the Bayesian model there is no need for randomization testing, and (since we need only to search the original grid) even a naive search can be performed relatively quickly. We now present the Bayesian spatial scan statistic, and then compare it to the frequentist approach on the task of detecting simulated disease epidemics.

## 2 The Bayesian scan statistic

Here we consider the natural Bayesian extension of Kulldorff's scan statistic, moving from a Poisson to a conjugate Gamma-Poisson model. Bayesian Gamma-Poisson models are a common representation for count data in epidemiology, and have been used in disease mapping by Clayton and Kaldor [7], Mollié [8], and others. In disease mapping, the effect of the Gamma prior is to produce a spatially smoothed map of disease rates; here we instead focus on computing the posterior probabilities, allowing us to determine the likelihood that an outbreak has occurred, and to estimate the location and size of potential outbreaks.

For the Bayesian spatial scan, as in the frequentist approach, we wish to compare the null hypothesis $H_0$ of no clusters to the set of alternative hypotheses $H_1(S)$, each representing a cluster in some region $S$. As before, we assume Poisson likelihoods, $c_i \sim \text{Po}(qb_i)$. The difference is that we assume a hierarchical Bayesian model where the disease rates $q_{in}$, $q_{out}$, and $q_{all}$ are themselves drawn from Gamma distributions. Thus, under the null hypothesis $H_0$, we have $q = q_{all}$ for all $s_i \in G$, where $q_{all} \sim \text{Ga}(\alpha_{all}, \beta_{all})$. Under the alternative hypothesis $H_1(S)$, we have $q = q_{in}$ for all $s_i \in S$ and $q = q_{out}$ for all $s_i \in G - S$, where we independently draw $q_{in} \sim \text{Ga}(\alpha_{in}, \beta_{in})$ and $q_{out} \sim \text{Ga}(\alpha_{out}, \beta_{out})$. We discuss how the $\alpha$ and $\beta$ priors are chosen below. From this model, we can compute the posterior probabilities $P(H_1(S)|D)$ of an outbreak in each region $S$, and the probability $P(H_0|D)$ that no outbreak has occurred, given dataset $D$: $P(H_0 | D) = \frac{P(D|H_0)P(H_0)}{P(D)}$ and $P(H_1(S)|D) = \frac{P(D|H_1(S))P(H_1(S))}{P(D)}$, where $P(D) = P(D|H_0)P(H_0) + \sum_S P(D|H_1(S))P(H_1(S))$. We discuss the choice of prior probabilities $P(H_0)$ and $P(H_1(S))$ below. To compute the marginal likelihood of the data given each hypothesis, we must integrate over all possible values of the parameters ($q_{in}$, $q_{out}$, $q_{all}$) weighted by their respective probabilities. Since we have chosen a conjugate prior, we can easily obtain a closed-form solution for these likelihoods:

$$P(D|H_0) = \int P(q_{all} \sim \text{Ga}(\alpha_{all}, \beta_{all})) \prod_{s_i \in G} P(c_i \sim \text{Po}(q_{all}b_i)) \, dq_{all}$$

$$P(D|H_1(S)) = \int P(q_{in} \sim \text{Ga}(\alpha_{in}, \beta_{in})) \prod_{s_i \in S} P(c_i \sim \text{Po}(q_{in}b_i)) \, dq_{in}$$
$$\times \int P(q_{out} \sim \text{Ga}(\alpha_{out}, \beta_{out})) \prod_{s_i \in G-S} P(c_i \sim \text{Po}(q_{out}b_i)) \, dq_{out}$$

Now, computing the integral, and letting $C = \sum c_i$ and $B = \sum b_i$, we obtain:

$$\int P(q \sim \text{Ga}(\alpha, \beta)) \prod_{s_i} P(c_i \sim \text{Po}(qb_i)) \, dq = \int \frac{\beta^\alpha}{\Gamma(\alpha)} q^{\alpha-1} e^{-\beta q} \prod_{s_i} \frac{(qb_i)^{c_i} e^{-qb_i}}{(c_i)!} \, dq \propto$$
$$\frac{\beta^\alpha}{\Gamma(\alpha)} \int q^{\alpha-1} e^{-\beta q} q^{\sum c_i} e^{-q \sum b_i} \, dq = \frac{\beta^\alpha}{\Gamma(\alpha)} \int q^{\alpha+C-1} e^{-(\beta+B)q} \, dq = \frac{\beta^\alpha \, \Gamma(\alpha+C)}{(\beta+B)^{\alpha+C} \, \Gamma(\alpha)}$$

Thus we have the following expressions for the marginal likelihoods: $P(D | H_0) \propto \frac{(\beta_{all})^{\alpha_{all}} \Gamma(\alpha_{all}+C_{all})}{(\beta_{all}+B_{all})^{\alpha_{all}+C_{all}} \Gamma(\alpha_{all})}$, and $P(D | H_1(S)) \propto \frac{(\beta_{in})^{\alpha_{in}} \Gamma(\alpha_{in}+C_{in})}{(\beta_{in}+B_{in})^{\alpha_{in}+C_{in}} \Gamma(\alpha_{in})} \times \frac{(\beta_{out})^{\alpha_{out}} \Gamma(\alpha_{out}+C_{out})}{(\beta_{out}+B_{out})^{\alpha_{out}+C_{out}} \Gamma(\alpha_{out})}$.

The Bayesian spatial scan statistic can be computed simply by first calculating the score $P(D\,|\,H_1(S))P(H_1(S))$ for each spatial region $S$, maintaining a list of regions ordered by score. We then calculate $P(D\,|\,H_0)P(H_0)$, and add this to the sum of all region scores, obtaining the probability of the data $P(D)$. Finally, we can compute the posterior probability $P(H_1(S)\,|\,D) = \frac{P(D\,|\,H_1(S))P(H_1(S))}{P(D)}$ for each region, as well as $P(H_0\,|\,D) = \frac{P(D\,|\,H_0)P(H_0)}{P(D)}$. Then we can return all regions with non-negligible posterior probabilities, the posterior probability of each, and the overall probability of an outbreak. Note that no randomization testing is necessary, and thus overall complexity is proportional to number of regions searched, e.g. $O(N^4)$ for searching over axis-aligned rectangles in an $N \times N$ grid.

## 2.1 Choosing priors

One of the most challenging tasks in any Bayesian analysis is the choice of priors. For any region $S$ that we examine, we must have values of the parameter priors $\alpha_{in}(S)$, $\beta_{in}(S)$, $\alpha_{out}(S)$, and $\beta_{out}(S)$, as well as the region prior probability $P(H_1(S))$. We must also choose the global parameter priors $\alpha_{all}$ and $\beta_{all}$, as well as the "no outbreak" prior $P(H_0)$.

Here we consider the simple case of a uniform region prior, with a known prior probability of an outbreak $P_1$. In other words, if there is an outbreak, it is assumed to be equally likely to occur in any spatial region. Thus we have $P(H_0) = 1 - P_1$, and $P(H_1(S)) = \frac{P_1}{N_{reg}}$, where $N_{reg}$ is the total number of regions searched. The parameter $P_1$ can be obtained from historical data, estimated by human experts, or can simply be used to tune the sensitivity and specificity of the algorithm. The model can also be easily adapted to a non-uniform region prior, taking into account our prior beliefs about the size and shape of outbreaks.

For the parameter priors, we assume that we have access to a large number of days of past data, during which no outbreaks are known to have occurred. We can then obtain estimated values of the parameter priors under the null hypothesis by matching the moments of each Gamma distribution to their historical values. In other words, we set the expectation and variance of the Gamma distribution $\mathrm{Ga}(\alpha_{all}, \beta_{all})$ to the sample expectation and variance of $\frac{C_{all}}{B_{all}}$ observed in past data: $\frac{\alpha_{all}}{\beta_{all}} = \mathrm{E}_{sample}\left[\frac{C_{all}}{B_{all}}\right]$, and $\frac{\alpha_{all}}{\beta_{all}^2} = \mathrm{Var}_{sample}\left[\frac{C_{all}}{B_{all}}\right]$. Solving for $\alpha_{all}$ and $\beta_{all}$, we obtain $\alpha_{all} = \frac{\left(\mathrm{E}_{sample}\left[\frac{C_{all}}{B_{all}}\right]\right)^2}{\mathrm{Var}_{sample}\left[\frac{C_{all}}{B_{all}}\right]}$ and $\beta_{all} = \frac{\mathrm{E}_{sample}\left[\frac{C_{all}}{B_{all}}\right]}{\mathrm{Var}_{sample}\left[\frac{C_{all}}{B_{all}}\right]}$.

The calculation of priors $\alpha_{in}(S)$, $\beta_{in}(S)$, $\alpha_{out}(S)$, and $\beta_{out}(S)$ is identical except for two differences: first, we must condition on the region $S$, and second, we must assume the alternative hypothesis $H_1(S)$ rather than the null hypothesis $H_0$. Repeating the above derivation for the "out" parameters, we obtain $\alpha_{out}(S) = \frac{\left(\mathrm{E}_{sample}\left[\frac{C_{out}(S)}{B_{out}(S)}\right]\right)^2}{\mathrm{Var}_{sample}\left[\frac{C_{out}(S)}{B_{out}(S)}\right]}$ and $\beta_{out}(S) = \frac{\mathrm{E}_{sample}\left[\frac{C_{out}(S)}{B_{out}(S)}\right]}{\mathrm{Var}_{sample}\left[\frac{C_{out}(S)}{B_{out}(S)}\right]}$, where $C_{out}(S)$ and $B_{out}(S)$ are respectively the total count $\sum_{G-S} c_i$ and total baseline $\sum_{G-S} b_i$ outside the region. Note that an outbreak in some region $S$ does not affect the disease rate outside region $S$. Thus we can use the same values of $\alpha_{out}(S)$ and $\beta_{out}(S)$ whether we are assuming the null hypothesis $H_0$ or the alternative hypothesis $H_1(S)$.

On the other hand, the effect of an outbreak inside region $S$ must be taken into account when computing $\alpha_{in}(S)$ and $\beta_{in}(S)$; since we assume that no outbreak has occurred in the past data, we cannot just use the sample mean and variance, but must consider what we expect these quantities to be in the event of an outbreak. We assume that the outbreak will increase $q_{in}$ by a multiplicative factor $m$, thus multiplying the mean and variance of $\frac{C_{in}}{B_{in}}$ by $m$. To account for this in the Gamma distribution $\mathrm{Ga}(\alpha_{in}, \beta_{in})$, we multiply $\alpha_{in}$ by $m$ while leaving $\beta_{in}$ unchanged. Thus we have $\alpha_{in}(S) = m\frac{\left(\mathrm{E}_{sample}\left[\frac{C_{in}(S)}{B_{in}(S)}\right]\right)^2}{\mathrm{Var}_{sample}\left[\frac{C_{in}(S)}{B_{in}(S)}\right]}$ and $\beta_{in}(S) = \frac{\mathrm{E}_{sample}\left[\frac{C_{in}(S)}{B_{in}(S)}\right]}{\mathrm{Var}_{sample}\left[\frac{C_{in}(S)}{B_{in}(S)}\right]}$,

where $C_{in}(S) = \sum_S c_i$ and $B_{in}(S) = \sum_S b_i$. Since we typically do not know the exact value of $m$, here we use a discretized uniform distribution for $m$, ranging from $m = 1 \dots 3$ at intervals of 0.2. Then scores can be calculated by averaging likelihoods over the distribution of $m$.

Finally, we consider how to deal with the case where the past values of the counts and baselines are not given. In this "blind Bayesian" (BBayes) case, we assume that counts are randomly generated under the null hypothesis $c_i \sim \text{Po}(q_0 b_i)$, where $q_0$ is the expected ratio of count to baseline under the null (for example, $q_0 = 1$ if baselines are obtained by estimating the expected value of the count). Under this simple assumption, we can easily compute the expectation and variance of the ratio of count to baseline under the null hypothesis: $\text{E}\left[\frac{C}{B}\right] = \frac{\text{E}[\text{Po}(q_0 B)]}{B} = \frac{q_0 B}{B} = q_0$, and $\text{Var}\left[\frac{C}{B}\right] = \frac{\text{Var}[\text{Po}(q_0 B)]}{B^2} = \frac{q_0 B}{B^2} = \frac{q_0}{B}$. Thus we have $\alpha = q_0 B$ and $\beta = B$ under the null hypothesis. This gives us $\alpha_{all} = q_0 B_{all}$, $\beta_{all} = B_{all}$, $\alpha_{out}(S) = q_0 B_{out}(S)$, $\beta_{out}(S) = B_{out}(S)$, $\alpha_{in}(S) = m q_0 B_{in}(S)$, and $\beta_{in}(S) = B_{in}(S)$. We can use a uniform distribution for $m$ as before. In our empirical evaluation below, we consider both the Bayes and BBayes methods of generating parameter priors.

## 3  Results: detection power

We evaluated the Bayesian and frequentist methods on two types of simulated respiratory outbreaks, injected into real Emergency Department and over-the-counter drug sales data for Allegheny County, Pennsylvania. All data were aggregated to the zip code level to ensure anonymity, giving the daily counts of respiratory ED cases and sales of OTC cough and cold medication in each of 88 zip codes for one year. The baseline (expected count) for each zip code was estimated using the mean count of the previous 28 days. Zip code centroids were mapped to a $16 \times 16$ grid, and all rectangles up to $8 \times 8$ were examined. We first considered simulated aerosol releases of inhalational anthrax (e.g. from a bioterrorist attack), generated by the Bayesian Aerosol Release Detector, or BARD [9]. The BARD simulator uses a Bayesian network model to determine the number of spores inhaled by individuals in affected areas, the resulting number and severity of anthrax cases, and the resulting number of respiratory ED cases on each day of the outbreak in each affected zip code. Our second type of outbreak was a simulated "Fictional Linear Onset Outbreak" (or "FLOO"), as in [10]. A FLOO$(\Delta, T)$ outbreak is a simple simulated outbreak with duration $T$, which generates $t\Delta$ cases in each affected zip code on day $t$ of the outbreak $(0 < t \leq T/2)$, then generates $T\Delta/2$ cases per day for the remainder of the outbreak. Thus we have an outbreak where the number of cases ramps up linearly and then levels off. While this is clearly a less realistic outbreak than the BARD-simulated anthrax attack, it does have several advantages: most importantly, it allows us to precisely control the slope of the outbreak curve and examine how this affects our methods' detection ability.

To test detection power, a semi-synthetic testing framework similar to [10] was used: we first run our spatial scan statistic for each day of the last nine months of the year (the first three months are used only to estimate baselines and priors), and obtain the score $F^*$ for each day. Then for each outbreak we wish to test, we inject that outbreak into the data, and obtain the score $F^*(t)$ for each day $t$ of the outbreak. By finding the proportion of baseline days with scores higher than $F^*(t)$, we can determine the proportion of false positives we would have to accept to detect the outbreak on day $t$. This allows us to compute, for any given level of false positives, what proportion of outbreaks can be detected, and the mean number of days to detection. We compare three methods of computing the score $F^*$: the frequentist method ($F^*$ is the maximum likelihood ratio $F(S)$ over all regions $S$), the Bayesian maximum method ($F^*$ is the maximum posterior probability $P(H_1(S) \mid D)$ over all regions $S$), and the Bayesian total method ($F^*$ is the sum of posterior probabilities $P(H_1(S) \mid D)$ over all regions $S$, i.e. total posterior probability of an outbreak). For the two Bayesian methods, we consider both Bayes and BBayes methods for calculating priors, thus giving us a total of five methods to compare (frequentist, Bayes_max, BBayes_max, Bayes_tot, BBayes_tot). In Table 1, we compare these methods with respect to proportion of outbreaks detected and

Table 1: Days to detect and proportion of outbreaks detected, 1 false positive/month

| method | FLOO_ED (4,14) | FLOO_ED (2,20) | FLOO_ED (1,20) | BARD_ED (.125) | BARD_ED (.016) | FLOO_OTC (40,14) | FLOO_OTC (25,20) |
|---|---|---|---|---|---|---|---|
| frequentist | 1.859 (100%) | 3.324 (100%) | 6.122 (96%) | 1.733 (100%) | 3.925 (88%) | 3.582 (100%) | 5.393 (100%) |
| Bayes_max | 1.740 (100%) | 2.875 (100%) | 5.043 (100%) | **1.600** **(100%)** | 3.755 (88%) | 5.455 (63%) | 7.588 (79%) |
| BBayes_max | **1.683** **(100%)** | **2.848** **(100%)** | **4.984** **(100%)** | **1.600** **(100%)** | **3.698** **(88%)** | 5.164 (65%) | 7.035 (77%) |
| Bayes_tot | 1.882 (100%) | 3.195 (100%) | 5.777 (100%) | 1.633 (100%) | 3.811 (88%) | **3.475** **(100%)** | **5.195** **(100%)** |
| BBayes_tot | 1.840 (100%) | 3.180 (100%) | 5.672 (100%) | 1.617 (100%) | 3.792 (88%) | 4.380 (100%) | 6.929 (99%) |

mean number of days to detect, at a false positive rate of 1/month. Methods were evaluated on seven types of simulated outbreaks: three FLOO outbreaks on ED data, two FLOO outbreaks on OTC data, and two BARD outbreaks (with different amounts of anthrax release) on ED data. For each outbreak type, each method's performance was averaged over 100 or 250 simulated outbreaks for BARD or FLOO respectively.

In Table 1, we observe very different results for the ED and OTC datasets. For the five runs on ED data, all four Bayesian methods consistently detected outbreaks faster than the frequentist method. This difference was most evident for the more slowly growing (harder to detect) outbreaks, especially FLOO(1,20). Across all ED outbreaks, the Bayesian methods showed an average improvement of between 0.13 days (Bayes_tot) and 0.43 days (BBayes_max) as compared to the frequentist approach; "max" methods performed substantially better than "tot" methods, and "BBayes" methods performed slightly better than "Bayes" methods. For the two runs on OTC data, on the other hand, most of the Bayesian methods performed much worse (over 1 day slower) than the frequentist method. The exception was the Bayes_tot method, which again outperformed the frequentist method by an average of 0.15 days. We believe that the main reason for these differing results is that the OTC data is much noisier than the ED data, and exhibits much stronger seasonal trends. As a result, our baseline estimates (using mean of the previous 28 days) are reasonably accurate for ED, but for OTC the baseline estimates will lag behind the seasonal trends (and thus, underestimate the expected counts for increasing trends and overestimate for decreasing trends). The BBayes methods, which assume $E[C/B] = 1$ and thus rely heavily on the accuracy of baseline estimates, are not reasonable for OTC. On the other hand, the Bayes methods (which instead learn the priors from previous counts and baselines) can adjust for consistent misestimation of baselines and thus more accurately account for these seasonal trends. The "max" methods perform badly on the OTC data because a large number of baseline days have posterior probabilities close to 1; in this case, the maximum region posterior varies wildly from day to day, depending on how much of the total probability is assigned to a single region, and is not a reliable measure of whether an outbreak has occurred. The total posterior probability of an outbreak, on the other hand, will still be higher for outbreak than non-outbreak days, so the "tot" methods can perform well on OTC as well as ED data. Thus, our main result is that the Bayes_tot method, which infers baselines from past counts and uses total posterior probability of an outbreak to decide when to sound the alarm, consistently outperforms the frequentist method for both ED and OTC datasets.

## 4 Results: computation time

As noted above, the Bayesian spatial scan must search over all rectangular regions for the original grid only, while the frequentist scan (in order to calculate statistical significance by randomization) must also search over all rectangular regions for a large number (typically $R = 1000$) of replica grids. Thus, as long as the search time per region is comparable for the Bayesian and frequentist methods, we expect the Bayesian approach to be approximately 1000x faster. In Table 2, we compare the run times of the Bayes, BBayes, and frequen-

Table 2: Comparison of run times for varying grid size $N$

| method | $N = 16$ | $N = 32$ | $N = 64$ | $N = 128$ | $N = 256$ |
|---|---|---|---|---|---|
| Bayes (naive) | 0.7 sec | 10.8 sec | 2.8 min | 44 min | 12 hrs |
| BBayes (naive) | 0.6 sec | 9.3 sec | 2.4 min | 37 min | 10 hrs |
| frequentist (naive) | 12 min | 2.9 hrs | 49 hrs | $\sim$31 days | $\sim$500 days |
| frequentist (fast) | 20 sec | 1.8 min | 10.7 min | 77 min | 10 hrs |

tist methods for searching a single grid and calculating significance ($p$-values or posterior probabilities for the frequentist and Bayesian methods respectively), as a function of the grid size $N$. All rectangles up to size $N/2$ were searched, and for the frequentist method $R = 1000$ replications were performed. The results confirm our intuition: the Bayesian methods are 900-1200x faster than the frequentist approach, for all values of $N$ tested. However, the frequentist approach can be accelerated dramatically using our "fast spatial scan" algorithm [2], a multiresolution search method which can find the highest scoring region of a grid while searching only a small subset of regions. Comparing the fast spatial scan to the Bayesian approach, we see that the fast spatial scan is slower than the Bayesian method for grid sizes up to $N = 128$, but slightly faster for $N = 256$. Thus we now have two options for making the spatial scan statistic computationally feasible for large grid sizes: to use the fast spatial scan to speed up the frequentist scan statistic, or to use the Bayesian scan statistics framework (in which case the naive algorithm is typically fast enough). For even larger grid sizes, it may be possible to extend the fast spatial scan to the Bayesian approach: this would give us the best of both worlds, searching only one grid, and using a fast algorithm to do so. We are currently investigating this potentially useful synthesis.

## 5   Discussion

We have presented a Bayesian spatial scan statistic, and demonstrated several ways in which this method is preferable to the standard (frequentist) scan statistics approach. In Section 3, we demonstrated that the Bayesian method, with a relatively non-informative prior distribution, consistently outperforms the frequentist method with respect to detection power. Since the Bayesian framework allows us to easily incorporate prior information about size, shape, and impact of an outbreak, it is likely that we can achieve even better detection performance using more informative priors, e.g. obtained from experts in the domain. In Section 4, we demonstrated that the Bayesian spatial scan can be computed in much less time than the frequentist method, since randomization testing is unnecessary. This allows us to search large grid sizes using a naive search algorithm, and even larger grids might be searched by extending the fast spatial scan to the Bayesian framework.

We now consider three other arguments for use of the Bayesian spatial scan. First, the Bayesian method has easily interpretable results: it outputs the posterior probability that an outbreak has occurred, and the distribution of this probability over possible outbreak regions. This makes it easy for a user (e.g. public health official) to decide whether to investigate each potential outbreak based on the costs of false positives and false negatives; this type of decision analysis cannot be done easily in the frequentist framework. Another useful result of the Bayesian method is that we can compute a "map" of the posterior probabilities of an outbreak in each grid cell, by summing the posterior probabilities $P(H_1(S)|D)$ of all regions containing that cell. This technique allows us to deal with the case where the posterior probability mass is spread among many regions, by observing cells which are common to most or all of these regions. We give an example of such a map below:

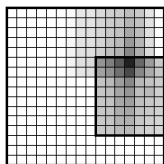

Figure 1: Output of Bayesian spatial scan on baseline OTC data, 1/30/05. Cell shading is based on posterior probability of an outbreak in that cell, ranging from white (0%) to black (100%). The bold rectangle represents the most likely region (posterior probability 12.27%) and the darkest cell is the most likely cell (total posterior probability 86.57%). Total posterior probability of an outbreak is 86.61%.

Second, calibration of the Bayesian statistic is easier than calibration of the frequentist statistic. As noted above, it is simple to adjust the sensitivity and specificity of the Bayesian method by setting the prior probability of an outbreak $P_1$, and then we can "sound the alarm" whenever posterior probability of an outbreak exceeds some threshold. In the frequentist method, on the other hand, many regions in the baseline data have sufficiently high likelihood ratios that no replicas beat the original grid; thus we cannot distinguish the $p$-values of outbreak and non-outbreak days. While one alternative is to "sound the alarm" when the likelihood ratio is above some threshold (rather than when $p$-value is below some threshold), this is technically incorrect: because the baselines for each day of data are different, the distribution of region scores under the null hypothesis will also differ from day to day, and thus days with higher likelihood ratios do not necessarily have lower $p$-values. Third, we argue that it is easier to combine evidence from multiple detectors within the Bayesian framework, i.e. by modeling the joint probability distribution. We are in the process of examining Bayesian detectors which look simultaneously at the day's Emergency Department records and over-the-counter drug sales in order to detect emerging clusters, and we believe that combination of detectors is an important area for future research.

In conclusion, we note that, though both Bayesian modeling [7-8] and (frequentist) spatial scanning [3-4] are common in the spatial statistics literature, this is (to the best of our knowledge) the first model which combines the two techniques into a single framework. In fact, very little work exists on Bayesian methods for spatial cluster detection. One notable exception is the literature on spatial cluster modeling [11-12], which attempts to infer the location of cluster centers by inferring parameters of a Bayesian process model. Our work differs from these methods both in its computational tractability (their models typically have no closed form solution, so computationally expensive MCMC approximations are used) and its easy interpretability (their models give no indication as to statistical significance or posterior probability of clusters found). Thus we believe that this is the first Bayesian spatial cluster detection method which is powerful and useful, yet computationally tractable. We are currently running the Bayesian and frequentist scan statistics on daily OTC sales data from over 10000 stores, searching for emerging disease outbreaks on a daily basis nationwide. Additionally, we are working to extend the Bayesian statistic to fMRI data, with the goal of discovering regions of brain activity corresponding to given cognitive tasks [13, 6]. We believe that the Bayesian approach has the potential to improve both speed and detection power of the spatial scan in this domain as well.

## References

[1] M. Kulldorff. 1999. Spatial scan statistics: models, calculations, and applications. In J. Glaz and M. Balakrishnan, eds., *Scan Statistics and Applications*, Birkhauser, 303-322.

[2] D. B. Neill and A. W. Moore. 2004. Rapid detection of significant spatial clusters. In *Proc. 10th ACM SIGKDD Intl. Conf. on Knowledge Discovery and Data Mining*, 256-265.

[3] M. Kulldorff and N. Nagarwalla. 1995. Spatial disease clusters: detection and inference. *Statistics in Medicine* **14**, 799-810.

[4] M. Kulldorff. 1997. A spatial scan statistic. *Communications in Statistics: Theory and Methods* **26**(6), 1481-1496.

[5] D. B. Neill and A. W. Moore. 2004. A fast multi-resolution method for detection of significant spatial disease clusters. In *Advances in Neural Information Processing Systems* **16**, 651-658.

[6] D. B. Neill, A. W. Moore, F. Pereira, and T. Mitchell. 2005. Detecting significant multidimensional spatial clusters. In *Advances in Neural Information Processing Systems* **17**, 969-976.

[7] D. G. Clayton and J. Kaldor. 1987. Empirical Bayes estimates of age-standardized relative risks for use in disease mapping. *Biometrics* **43**, 671-681.

[8] A. Mollié. 1999. Bayesian and empirical Bayes approaches to disease mapping. In A. B. Lawson, et al., eds. *Disease Mapping and Risk Assessment for Public Health*. Wiley, Chichester.

[9] W. Hogan, G. Cooper, M. Wagner, and G. Wallstrom. 2004. A Bayesian anthrax aerosol release detector. Technical Report, RODS Laboratory, University of Pittsburgh.

[10] D. B. Neill, A. W. Moore, M. Sabhnani, and K. Daniel. 2005. Detection of emerging space-time clusters. In *Proc. 11th ACM SIGKDD Intl. Conf. on Knowledge Discovery and Data Mining*.

[11] R. E. Gangnon and M. K. Clayton. 2000. Bayesian detection and modeling of spatial disease clustering. *Biometrics* **56**, 922-935.

[12] A. B. Lawson and D. G. T. Denison, eds. 2002. *Spatial Cluster Modelling*. Chapman & Hall/CRC, Boca Raton, FL.

[13] X. Wang, R. Hutchinson, and T. Mitchell. 2004. Training fMRI classifiers to detect cognitive states across multiple human subjects. In *Advances in Neural Information Processing Systems* **16**, 709-716.
